# Spike Feature Extraction Using Informative Samples

**Zhi Yang, Qi Zhao and Wentai Liu**
School of Engineering
University of California at Santa Cruz
1156 High Street, Santa Cruz, CA 95064
{yangzhi, zhaoqi, wentai}@soe.ucsc.edu

## Abstract

This paper presents a spike feature extraction algorithm that targets real-time spike sorting and facilitates miniaturized microchip implementation. The proposed algorithm has been evaluated on synthesized waveforms and experimentally recorded sequences. When compared with many spike sorting approaches our algorithm demonstrates improved speed, accuracy and allows unsupervised execution. A preliminary hardware implementation has been realized using an integrated microchip interfaced with a personal computer.

## 1  Introduction

Real-time extraction of information from composite neural recordings is a significant challenge in neural interfacing. Developing integrated circuit (IC) to enable portable and implantable systems is important to allow the study of complex behavior in neuroscience experiments, closed loop deep brain stimulation, and cortical controlled neuromuscular prostheses. In order for a spike feature extraction algorithm to be functional as a small device with real-time low-latency processing and low power operation it must be efficient in both computation and IC implementation.

Implementing spike sorting before data telemetry offers many significant advantages. Spike feature extraction provides the necessary information required to sort spikes from raw sampled data. With this information each spike event can be represented by its unique features and firing time, resulting in significant data compression. A data transceiver designed with the current semiconductor technology can simultaneously support a large number of recording channels for a microchip implementation to extract the spike feature. System integration using wireless power telemetry or a rechargeable battery as well as wireless data telemetry removes the need for tethering wires. As a result, a fully wireless operation would relieve the subjects overall stress factor and allow them to move freely in their natural environment.

Frequently used spike feature extraction algorithms include principal component analysis (PCA) [1], bayesian algorithm [2], template matching [3], wavelets [4] and independent component analysis (ICA) [5], which demand significant computation. Efforts to improve the efficiency of these algorithms have been reported, however, these efforts relied on either over simplified functionality or bulky hardware systems that consume excessive power.

In part, complex algorithm procedures are applied to mediate the effects of noise and distortion in the recording process. The associated noise includes ion channel noise, activities from distant neurons, field potentials, thermal noise and circuit noise. Significant sampling distortion is also present since it is unrealistic to synchronize the sampling clock with individual recorded spikes.

This paper reports a new spike feature extraction algorithm which is suitable for real-time spike sorting and enables integrated microchip implementation.

## 2   Related Work

### 2.1   PCA Based Spike Feature Extraction

PCA is a feature extraction algorithm widely employed for spike sorting. It uses correlation between samples and computes the vectors capturing the maximal variance. PCA algorithm performs well given a strong correlation between samples by reporting relevant features. However, recorded spikes are usually corrupted by large low frequency noise and distortion, which blur sample correlation and compromise the quality of the estimated covariance matrix and its eigenvectors. As a result, PCA may fail to resolve spike clusters in noisy recordings.

### 2.2   Variable Selection Techniques

As a complementary approach to dimensionality reduction algorithms, Jolliffe discussed a general feature extraction algorithm based on a subset of samples in the classic work [6]. This concept requires only a subset of samples containing the necessary information to cluster the data; as opposed to using all of the samples. These informative samples are especially useful in the presence of single prominent sample set.

There are two challenges facing a sample selection algorithm. The first challenge is the computational burden to select informative samples. If the training procedure is as complicated as suggested in [6], it would prohibit microchip implementation for implant purposes. The power and area are the primary problems with the microchip implementation of other spike feature extraction algorithms. The second challenge is the availability of localized features. Improved performance compared to PCA is unlikely if localized features are not prominent.

### 2.3   Our Approach

We have developed a spike feature extraction algorithm based on informative samples. The theoretical framework includes neuronal geometry signatures, noise shaping, and informative sample selection. By evaluating neuronal geometry signatures with the compartment model, we find that high frequency signal spectrum may contain useful information to differentiate neurons. Studying the noise properties has revealed that a frequency shaping filter can be used to boost the SNR. The sample selection technique using estimated entropy identifies informative samples for sorting spikes. In addition, a preliminary IC implementation of the algorithm has been reported [7, 8] and further integrated onto a multi-channel neural recording IC with wireless telemetry [9].

## 3   Geometry Signatures, Noise and Sampling Distortion

### 3.1   Neuronal Geometry Signature

This section describes how neuronal geometry signatures contribute to the difference among similar waveforms. Assume that both the intra- and extra- fluids are neutral, the induced voltage waveform is

$$V(\overrightarrow{r_0}) = \int \frac{j_m(\overrightarrow{r}, t) dr}{4\pi\sigma_e |\overrightarrow{r} - \overrightarrow{r_0}|}, \tag{1}$$

where $j_m$ is the transmembrane current and $\sigma_e$ is the conductivity of the tissue environment; $\overrightarrow{r_0}$ and $\overrightarrow{r}$ represent the locations of the point electrode and the active membrane segments, respectively.

Since action potentials propagate slowly along the axonal branches of the cortex neurons (averaged $0.5m/sec - 2m/sec$ [10]), active membranes do not fire simultaneously. As a result, the detailed geometry of the underlying neuron influences the shape of spikes. Assuming that ionic channels are uniformly dotted on the active membranes within the recording radius of the electrode, the spike waveform is modeled as the convolution of the transmembrane current profile and an implicit geometry kernel function as

$$V(t) = \int j_m(\tau)W(t-\tau)d\tau, \tag{2}$$

where $W(t)$ is the geometry kernel function.

The recorded waveforms from neurons with similar ion channel populations can be very similar. A general spike sorting algorithm frequently fails to resolve such ambiguity and may report a single, large, spike cluster. The approach of differentiating associated kernel functions can be used to sort the similar spikes. Assume $W_1(t)$ and $W_2(t)$ as the geometry kernel functions of two neurons with the same ion channel population, the difference between the two spikes is

$$\triangle V(t) = \int j_m(\tau)[W_1(t-\tau) - W_2(t-\tau)]d\tau, \tag{3}$$

Small waveform differences appear if $\int (W_1(t) - W_2(t))dt \approx 0$. Intuitively, the condition means the waveforms are identical, ignoring the skew of the activation of membranes.

To differentiate the waveforms, we rewrite Eq. 3 in the frequency domain as

$$\mathcal{F}(\triangle V) = \mathcal{F}(j_m)\mathcal{F}(W_1 - W_2) \tag{4}$$

where $\mathcal{F}()$ denotes the fourier transform. The condition of $\int [W_1(t) - W_2(t)]dt \approx 0$ is equivalent to $\mathcal{F}(W_1 - W_2) \approx 0|_{f=0Hz}$, which implies that the waveform difference caused by the geometry kernel functions has small contribution at lower frequency spectrum. A more quantitative explanation can be given by studying the derivative of $\mathcal{F}(\triangle V)$ with respect to the frequency using Eq. 4

$$\frac{\partial \mathcal{F}(\triangle V)}{\partial f} = \frac{\partial \mathcal{F}(j_m)}{\partial f}\mathcal{F}(W_1 - W_2) + \mathcal{F}(j_m)\frac{\partial \mathcal{F}(W_1 - W_2)}{\partial f}, \tag{5}$$

where $f$ is frequency.

Note that $\mathcal{F}(j_m)$ is narrowly band limited signal and $\mathcal{F}(W_1 - W_2)$ serves as a notch frequency mask with a relative wider spectrum. The first term in Eq. 5 is attenuated by $\mathcal{F}(W_1 - W_2)$ within the dominant spectrum of $\mathcal{F}(j_m)$. Otherwise, appreciable waveform difference is expected according to Eq. 4. The second term in Eq. 5, on the other hand, exhibits a strong frequency dependency within the dominant spectrum of $\mathcal{F}(j_m)$. It can be expanded as

$$\mathcal{F}(j_m)\frac{\partial \mathcal{F}(W_1 - W_2)}{\partial f} \approx 2\pi \mathcal{F}(j_m) \int (W_1(t) - W_2(t))t\sin(2\pi ft)dt, \tag{6}$$

when kernel functions $W_i$ are symmetrical.

In summary, the waveform difference between similar neurons caused by geometry functions satisfies the following conditions

$$\begin{cases} \mathcal{F}(\triangle V) \approx 0|_{f=0Hz} \\ \frac{\partial \mathcal{F}(\triangle V)}{\partial f} \approx 4\pi^2 f\mathcal{F}(j_m) \int (W_1(t) - W_2(t))t\frac{\sin(2\pi ft)}{2\pi f}dt \propto f. \end{cases} \tag{7}$$

In Eq. 7, $\frac{\partial \mathcal{F}(\triangle V)}{\partial f}$ is linear to frequency $f$ at low frequency region, as $\frac{\sin(2\pi ft)}{2\pi ft} \approx 1$. The strong emphasis on frequency shows that $\mathcal{F}(\triangle V)$ exhibits a higher frequency spectrum. As a result, a frequency-shaping filter that emphasizes on high-frequency spectrum may help to differentiates kernel functions.

## 3.2  Noise and Sample Distortion

An estimated power spectrum of noise associated with recorded neural signal, where the dominance of low frequency noise is clear, is plotted in Figure 1. The noise profile is approximately fitted as

$$N(f) = N_{neu} + N_{e.e} + N_{1/f} + N_{therm} \approx N_{f_{c1}}(\frac{f_{c1}}{f})^\alpha + N_{therm}, \tag{8}$$

where $N_{neu}$ is the neuronal noise, $N_{e.e}$ is the electrode-electrolyte interface noise, $N_{1/f}$ is the flicker noise and $N_{therm}$ is the thermal noise. The low frequency noise is assumed to have profile following $f^{-\alpha}$.

Sampling distortion is unavoidable, since the neuron's firing is random and not synchronized with the sampling clock of the analog-to-digital converter(ADC). It can be reduced by either increasing the sampling frequency of the ADC or performing interpolation and alignment in the digital domain. Both approaches require additional power, computation and storage space, which are not favorable to microchip implementation. The sampling distortion is related to the slope of the spikes. In case a fast transition edge is sampled 4 times, the sampling distortion can be more than $10\%$ of the spike peak-to-peak magnitude. Considerable distortion is expected since "neural spikes" are, by definition, fast changing waveforms.

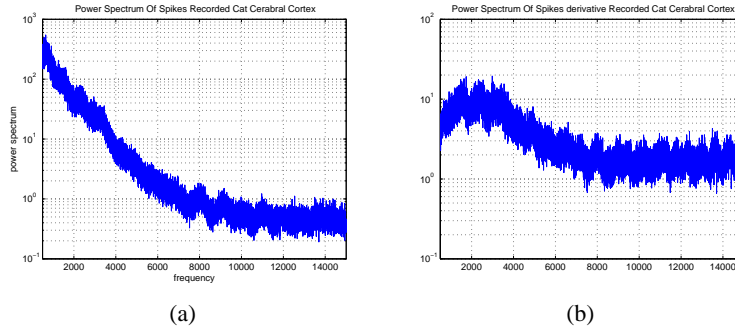

(a)                                                    (b)

Figure 1: noise properties of recordings from a cat cerebral cortex (500 Hz to 15K Hz); (a) noise power spectrum of raw data. (b) noise power spectrum of the derivative.

## 4 Sample Information

In order to use informative samples to sort spikes, it is necessary to quantify the information carried by individual spike samples. Intuitively, a sample is considered to be informative if the superimposed spikes can be classified into multiple clusters by evaluating that sample alone. The method used to quantify the sample information is outlined below.

---

Sample Information Estimation

---

Input: $M$ peak aligned spike segments $\{v_i, i = (1, M)\}$ with $N$ samples for each segment

Output: Information $info_j$ carried by spike samples $\{v_i(j), i = (1, M)\}$

- $j = 1$, construct one dimensional data set $X = \{v_i(j), i = (1, M)\}$
- Obtain a nested cluster configuration based on $X$
- Estimate the possibility $p_q$ that a spike being partitioned into the $q^{th}$ cluster. Use the entropy to estimate the information $info_j = -\sum_{p_q > p_0} p_q ln(p_q)$, where $p_0$ is a threshold of the cluster size.
- Repeat the procedures to a different sample, e.g. $j = j + 1$.

---

The computation required to accurately quantify the entropy of an underlying data set is typically high. However, only a rough estimation is required to select informative samples. Therefore, the amount of spikes to compute information can be reduced to a relatively small number, which should allow hardware implementation in terms of storage space and computation complexity. With the synthesized spike data we used, each sequence contains 3 neuronal sources with similar firing rate. As a result, the possible information score should be $0$, $\frac{1}{3}Ln(3) + \frac{2}{3}ln(1.5)$ or $Ln(3)$. When we increase the mount of training events to $M = 300$ the information scores approximately settle to the expected values, as shown in Figure 2.

Quantitative comparisons to investigate the existence of informative samples in noisy spikes have been done. Results using synthesized spikes with recordings from neocortex and basal ganglia [4] are shown in Figure 2. There are two clear observations. First, the amount of information carried by each sample varies, indicating a non-uniform signal-to-noise-plus-distortion-ratio. Second, it is necessary to create informative samples if due to severe noise, distortion and similarity of spike clusters, few of the samples is informative. As a constraint to create informative samples, the computation and storage space have to be feasible for microchip implementation.

## 5 Create Informative Samples Using Frequency Shaping Filter

As analyzed in Section 3, a frequency shaping filter can be used to manifest different geometry kernel functions, reduce noise and redistribute distortion among spike samples. Such a filter is

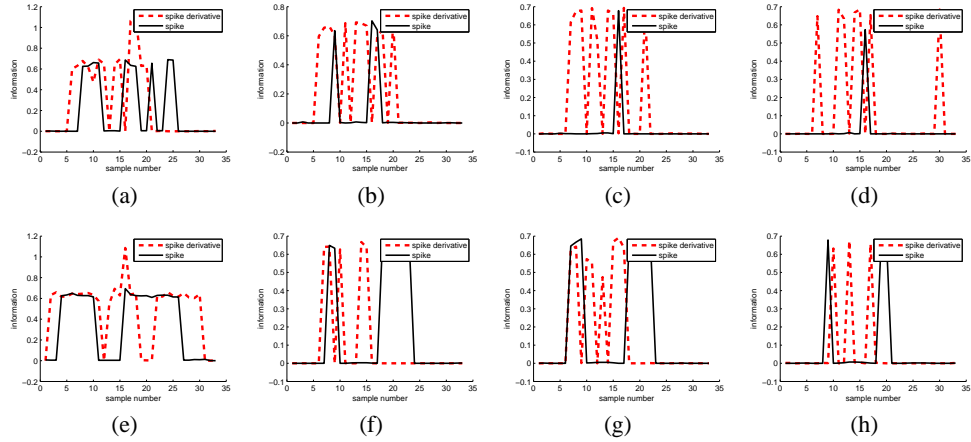

Figure 2: (a) - (h) information carried by samples from spikes and their derivatives. Horizontal axis is the sample number and vertical axis is the estimated entropy. The black solid line and red dotted line represent the sample information from spikes and their derivatives, respectively.

designed to boost high frequency spike features, which should be localized and less correlated if examined in time domain. In this section, we use derivative operation as an example to illustrate the usefulness of the frequency shaping filter, and further demonstrate that the filter creates additional informative samples.

In a discrete time spike sequence, the frequency response of taking derivative is

$$H(f) = 2e^{j\pi f/2}\sin(\pi f/f_s),$$ (9)

where $f_s$ is the sampling frequency of the ADC.

As shown in Section 3.1, the difference between neuron geometry kernel functions $W(t)$ of similar spikes is contained in the higher frequency components, which should be emphasized by derivative operation.

The noise power spectrum is modified by taking derivative. Intuitively, low frequency noise is reduced and the high frequency thermal noise is amplified, as shown in Figure 1 (b). The quantitative impact of the frequency shaping filter on noise is affected by the recording system and biological environment, and the typical values of $\alpha$ we observe vary around 2 within the signal band as shown in Figure 1. Use $\alpha = 2$ for illustration, the filter's influence on noise could be quantified by $\alpha$ Eq. 9

$$\lambda \approx \frac{f_{c1}f_{c2}}{2f_{spike}^2} \leq \frac{1}{2},$$ (10)

where $f_{c1}$ and $f_{c2}$ are the lower and higher corner frequencies of the digital filter, respectively. In case $\lambda$ is less than 1, SNR further increases, which favors spike sorting from the noise perspective.

The sampling distortion distribution among samples is altered after taking the derivative. In the original waveforms, samples close to peaks suffer less distortion compared with those in transition. After taking the derivative, samples initially suffering from large distortion become less distorted because $V''(t)$ in Eq. 2 has at least one zero crossing point during the transition. Quantitative experiments to demonstrate the creation of informative samples have been done.

A subset of results are shown in Figure 2 (a) - (h). In these data, the black solid lines represent information carried by the samples from spikes and the dotted red lines represent the derivatives. The spike data are 8 challenging sequences from [4]. They are compiled from recordings in the neocortex and basal ganglia with superimposed noise. All 8 sequences contain 3 neuronal sources.

During estimation of sample entropy, a mean shift classifier with a hierarchical merging procedure is being used to quantify the partition. Small clusters with events less than 5% are ignored. The corresponding feature extraction results using the most informative samples from spikes as well as their derivatives are shown in Figure 3 (a) - (h), which clearly presents a 3 cluster configuration.

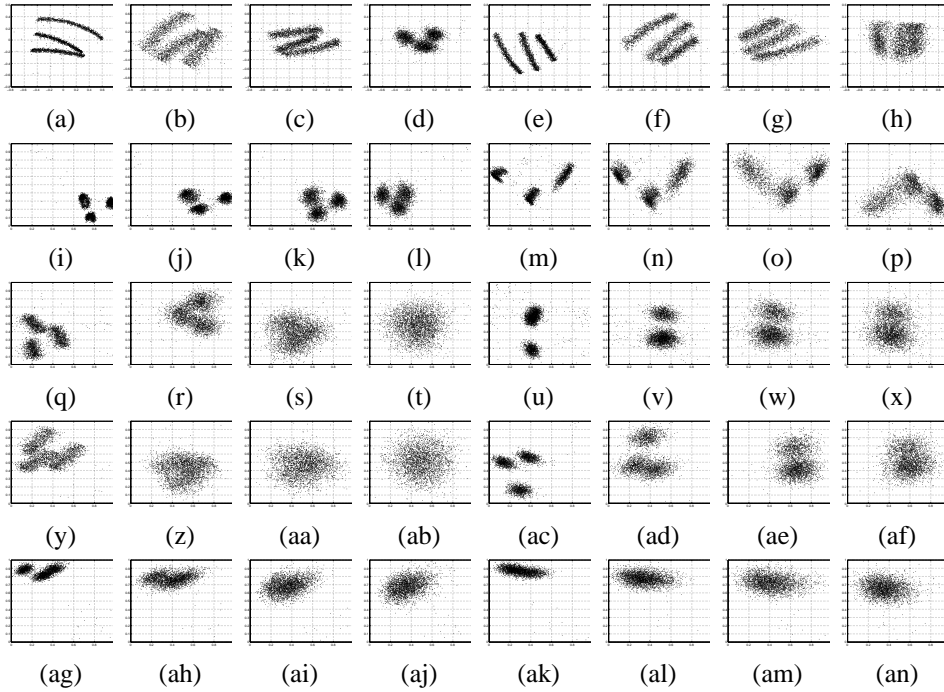

Figure 3: feature extraction results using the proposed algorithm and competing algorithms. (a) - (h) display the extracted features using the most informative samples of spikes and their derivatives (proposed). (i) - (p) display the extracted features using a subset of samples includes the peaks of the spike derivative and spike height (implemented on chip, proposed). (q) - (x) display the PCA based feature extraction. (y) - (af) display the wavelets based feature extraction. (ag) - (an) display spike peaks based feature extraction. (All the algorithms are tested without performing interpolation. Nonlinear energy operator (NEO) [11] is used as the spike detection algorithm. Overlapping spikes within 600 $\mu Sec$ are ignored. Haar wavelet is used to perform wavelets based feature extraction, and features are obtained from the variance peaks after the wavelet transform. Two dimensional features are projected from a higher dimensional space.)

Table 1: Accuracy comparison of using different spike feature extraction algorithms

| Sequence Number | 1 | 2 | 3 | 4 | 5 | 6 | 7 | 8 |
|---|---|---|---|---|---|---|---|---|
| Informative Samples | 97.8% | 97.8% | 97.8% | 97.0% | 98.0% | 99.2% | 96.6% | 92.0% |
| Hardware | 97.6% | 97.6% | 97.4% | 95.4% | 98.2% | 98.4% | 93.2% | 91.0% |
| PCA | 97.8% | 89.0% | 60.4% | 55.2% | 97.6% | 77.8% | 80.2% | 68.8% |
| Wavelets | 92.4% | 91.0% | 81.8% | 57.4% | 97.4% | 68.2% | 51.0% | 49.4% |
| Spike Peaks | 34.2% | 33.8% | 35.4% | 34.0% | 36.2% | 37.8% | 35.6% | 36.0% |

Note: Informative samples are harvested from both spikes and their derivatives. Hardware uses peaks of spikes and their derivatives. 3000 spikes each sequence from [4].

## 6   Experiments

Synthesized spike sequences used in Figure 2 are applied to compare the sorting accuracies of different approaches. Feature extraction using the pre-specified subset consists of the peaks of the spike derivative as well as the height of the original spike is shown in Figure 3 (i) - (p). Comparative feature extraction results using competing algorithms, e.g, PCA, wavelets, spike peaks and width are also shown in Figure 3. The extracted spike features are clustered on a PC [12]. About $5\%$ overlapping spikes are ignored to clearly quantify the performance of different spike feature extraction algorithms. The proposed feature extraction algorithm including the most informative samples (corresponding to Figure 3 (a) - (h)) achieves the highest accuracy ($97.0\%$). The hardware [9, 8]

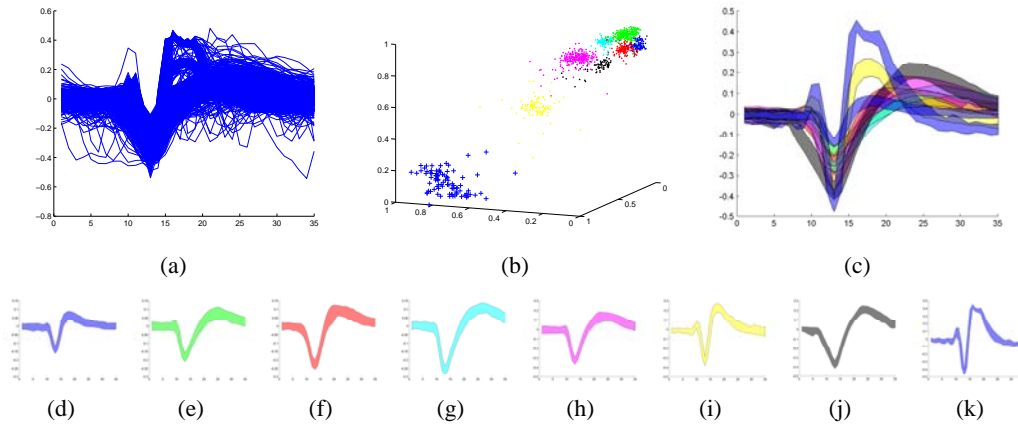

Figure 4: (a) recorded spikes from cat cerebral cortex are superimposed, (b) the extracted spike features using a subset of samples are plotted and grouped with a clustering algorithm implemented on PC. (c) the classified spike clusters are superimposed. (d) - (k) individual spike clusters superimposed in (c) are displayed. Spike clusters in (d) - (g) are plotted in a smaller vertical scale (-0.3, 0.15) compared with (h) - (j) in (-0.5, 0.3) and (k) in (-0.5, 0.5).

using the pre-specified subset gives similar accuracy (96.1%). The counterpart algorithms include PCA, wavelets and spike peaks and width give 78.4%, 73.6% and 35.4%, respectively. The sorting accuracy comparisons are listed in Table 1.

Animal sequences are collected to test the performance of the proposed algorithm. An example with overlapped spike clusters is selected for demonstration. The sequence is recorded from the cat cerebral cortex. The sorting results are displayed in Figure 4. In Figure 4 (a), the detected 1210 spikes are superimposed. Extracted spike features using the pre-specified subset of samples implemented on chip are shown in Figure 4 (b). The discrete points in feature space are grouped into 8 clusters with different colors using off-line clustering. Less than 10 % of noisy spikes and overlapping spikes are discarded, the rest are classified and plotted in Figure 4(c). To further quantify the validity of the classified spike clusters, superimposed clusters in Figure 4(c) are individually plotted in Figure 4(d)-(k).

The second example containing more than 4000 spikes recorded from a monkey is shown in Figure 5. In Figure 5 (a), detected spikes are superimposed. Extracted features using the pre-specified subset of informative samples are shown in Figure 5 (b). A zoom in of Figure 5 (b) is plotted in Figure 5 (c) to display the isolation quality of clusters in feature space. The corresponding PCA based feature extraction is shown in Figure 5 (d) as a comparison. The classified spike clusters using the pre-specified subset of informative samples are plotted in Figure 6 (a) - (e). Spike clusters plotted in Figure 6 (b), (c) and (d) resemble each other in shape and magnitude. To demonstrate that the informative samples based sorting does not over partitioning the data set, the derivatives of spike clusters plotted in Figure 6 (a) - (e) are also plotted in Figure 6 (f)-(j) with the same color indication. Clearly, Figure 6 (g), (h) and (i) present three well-differentiated waveform patterns in either peak-to-peak magnitude or shape.

## 7    Conclusion

A sample selection based spike feature extraction algorithm is reported in this paper. The theoretical framework includes neuronal geometry signatures, frequency shaping filter, and informative sample selection. Unlike PCA which uses correlated features, the sample selection algorithm focuses on localized and uncorrelated features which are strengthened by the frequency shaping filter. With simulated spike waveforms from a public data base, the algorithm demonstrates an improved sorting accuracy compared with many competing algorithms. The algorithm is designed for integrated microchip implementation and performing real-time spike sorting. A preliminary hardware implementation has been realized using an integrated circuit chip interfaced with a personal computer.

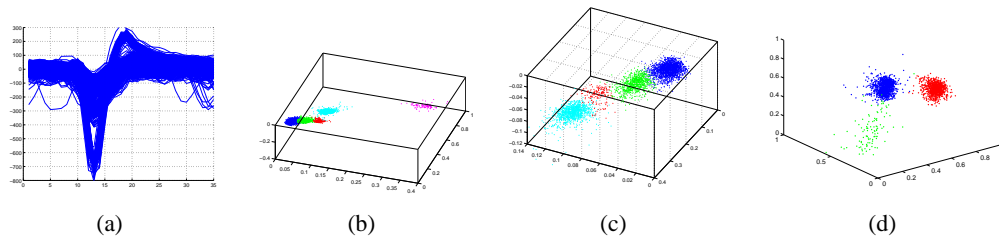

Figure 5: (a) detected spikes from a monkey, (b) extracted spike features using a subset of samples, (c) zoom in of (b) for better visualization; (d) extracted features using PCA.

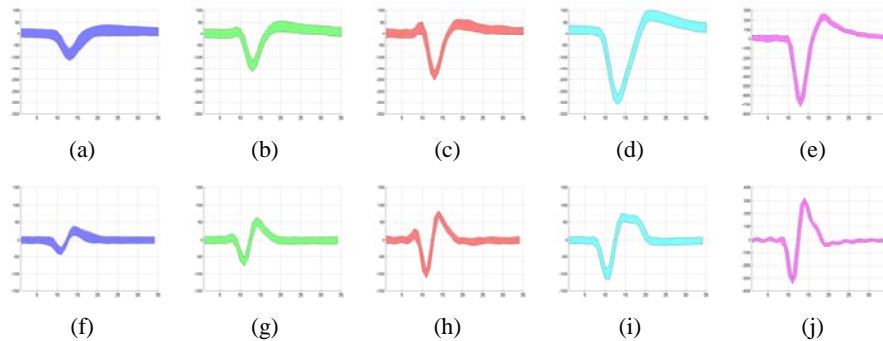

Figure 6: (a) - (e) the classified 5 clusters of the monkey sequence shown in Figure 5, (f)-(j) the derivative of the classified 5 clusters. The identity is indicated by color.

# References

[1] Zumsteg ZS, Kemere C, O'Driscoll S, Santhanam G, Ahmed RE, Shenoy KV, et al. Power feasibility of implantable digital spike sorting circuits for neural prosthetic systems. IEEE Trans Neural Syst Rehabil Eng. 2005 Sep;13(3):272–279.

[2] Lewicki MS. Bayesian modeling and classification of neural signals. Advances in NIPS. 1994;p. 590–597.

[3] Vargas-Irwin C, Donoghue JP. Automated spike sorting using density grid contour clustering and subtractive waveform decomposition. J Neurosci Methods. 2007;164(1).

[4] Quian Quiroga R, Nadasdy Z, Ben-Shaul Y. Unsupervised spike detection and sorting with wavelets and superparamagnetic clustering. Neural Comput. 2004 Aug;16(8):1661–1687.

[5] Takahashi S, Sakurai Y. Coding of spatial information by soma and dendrite of pyramidal cells in the hippocampal CA1 of behaving rats. Eur J Neurosci Methods. 2007 Oct;26(7):2033–2045.

[6] Jolliffe IT. Principal Component Analysys. New York: Springer-Verlag. 2002;.

[7] Yang Z, Chen T, Liu W. A neuron signature based spike feature extraction algorithm for on-chip implementation. to Appear in Proc 30th Ann Int Conf IEEE EMBS. 2008 Aug;p. 4237–4240.

[8] Chen T, Yang Z, Liu W, Chen L. NEUSORT2.0: a multiple-channel neural signal processor with systolic array buffer and channel-interleaving processing schedule. to appear Proc 30th Ann Int Conf IEEE EMBS. 2008 Aug;p. 6652–6656.

[9] Chae M, Liu W, Yang Z, Chen T, Kim J, Sivaprakasam M, et al. A 128 channel 6mW wireless neural recording IC with on-the-fly spike sorting and UWB transmitter. IEEE ISSCC 2008 Dig Tech Papers. 2008 Feb;7(6):241–261.

[10] Buzsaki G, Penttonen M, Nadasdy Z, Bragin A. Pattern and inhibition-dependent invasion of pyramidal cell dendrites by fast spikes in the hippocampus in vivo. Proc Natl Acad Sci USA. 1996 Sep;93(18):9921–9925.

[11] Kaiser JF. On a simple algorithm to calculate the energy of a signal. In Proc IEEE Int Conf Acoustic Speech and Signal Processing. 1990;p. 381–384.

[12] Yang Z, Zhao Q, Liu W. Neural signal classification using a simplified feature set with nonparametric clustering. to appear in Neurocomputing;.
